# Stochastic Relational Models for Large-scale Dyadic Data using MCMC

**Shenghuo Zhu**    **Kai Yu**    **Yihong Gong**
NEC Laboratories America, Cupertino, CA 95014, USA
{zsh, kyu, ygong}@sv.nec-labs.com

## Abstract

Stochastic relational models (SRMs) [15] provide a rich family of choices for learning and predicting dyadic data between two sets of entities. The models generalize matrix factorization to a supervised learning problem that utilizes attributes of entities in a hierarchical Bayesian framework. Previously variational Bayes inference was applied for SRMs, which is, however, not scalable when the size of either entity set grows to tens of thousands. In this paper, we introduce a Markov chain Monte Carlo (MCMC) algorithm for equivalent models of SRMs in order to scale the computation to very large dyadic data sets. Both superior scalability and predictive accuracy are demonstrated on a collaborative filtering problem, which involves tens of thousands users and half million items.

## 1  Stochastic Relational Models

Stochastic relational models (SRMs) [15] are generalizations of Gaussian process (GP) models [11] to the relational domain, where each observation is a dyadic datum, indexed by a pair of entities. They model dyadic data by a multiplicative interaction of two Gaussian process priors.

Let $\mathcal{U}$ be the feature representation (or index) space of a set of entities. A pair-wise similarity in $\mathcal{U}$ is given by a kernel (covariance) function $\Sigma : \mathcal{U} \times \mathcal{U} \rightarrow \mathbb{R}$. A Gaussian process (GP) defines a random function $f : \mathcal{U} \rightarrow \mathbb{R}$, whose distribution is characterized by a mean function and the covariance function $\Sigma$, denoted by $f \sim \mathcal{N}_{\infty}(0, \Sigma)$[1], where, for simplicity, we assume the mean to be the constant zero. GP complies with the intuition regarding the smoothness — if two entities $u_i$ and $u_j$ are similar according to $\Sigma$, then $f(u_i)$ and $f(u_j)$ are similar with a high probability.

A domain of dyadic data must involve another set of entities, let it be represented (or indexed) by $\mathcal{V}$. In a similar way, this entity set is associated with another kernel function $\Omega$. For example, in a typical collaborative filtering domain, $\mathcal{U}$ represents users while $\mathcal{V}$ represents items, then, $\Sigma$ measures the similarity between users and $\Omega$ measures the similarity between items.

Being the relation between a pair of entities from different sets, a dyadic variable $y$ is indexed by the product space $\mathcal{U} \times \mathcal{V}$. Then an SRM aims to model $y(u, v)$ by the following generative process,

**Model 1.** *The generative model of an SRM:*

1. *Draw kernel functions $\Sigma \sim \mathcal{IW}_{\infty}(\delta, \Sigma^{\circ})$, and $\Omega \sim \mathcal{IW}_{\infty}(\delta, \Omega^{\circ})$;*

2. *For $k = 1, \ldots, d$: draw random functions $f_k \sim \mathcal{N}_{\infty}(0, \Sigma)$, and $g_k \sim \mathcal{N}_{\infty}(0, \Omega)$;*

*3. For each pair $(u, v)$: draw $y(u, v) \sim p(y(u, v)|z(u, v), \gamma)$, where*

$$z(u, v) = \frac{1}{\sqrt{d}} \sum_{k=1}^{d} f_k(u) g_k(v) + b(u, v).$$

In this model, $\mathcal{IW}_\infty(\delta, \Sigma^\circ)$ and $\mathcal{IW}_\infty(\delta, \Omega^\circ)$ are *hyper priors*, whose details will be introduced later. $p(y|z, \gamma)$ is the problem-specific noise model. For example, it can follow a Gaussian noise distribution $y \sim \mathcal{N}_1(z, \gamma)$ if $y$ is numerical, or, a Bernoulli distribution if $y$ is binary. Function $b(u, v)$ is the bias function over the $\mathcal{U} \times \mathcal{V}$. For simplicity, we assume $b(u, v) = 0$.

In the limit $d \to \infty$, the model converges to a special case where $f_k$ and $g_k$ can be analytically marginalized out and $z$ becomes a Gaussian process $z \sim \mathcal{N}_\infty(0, \Sigma \otimes \Omega)$ [15], with the covariance between pairs being a tensor kernel

$$K\left((u_i, v_s), (u_j, v_t)\right) = \Sigma(u_i, u_j) \Omega(v_s, v_t).$$

In anther special case, if $\Sigma$ and $\Omega$ are both fixed to be Dirac delta functions, and $\mathcal{U}, \mathcal{V}$ are finite sets, it is easy to see that the model reduces to probabilistic matrix factorization.

The hyper prior $\mathcal{IW}_\infty(\delta, \Sigma^\circ)$ is called *inverted Wishart Process* that generalizes the finite $n$-dimensional inverted Wishart distribution [2]

$$\mathcal{IW}_n(\mathbf{\Sigma}|\delta, \mathbf{\Sigma}^\circ) \propto |\mathbf{\Sigma}|^{-\frac{1}{2}(\delta+2n)} \operatorname{etr}\left(-\frac{1}{2}\mathbf{\Sigma}^{-1}\mathbf{\Sigma}^\circ\right),$$

where $\delta$ is the degree-of-freedom parameter, and $\mathbf{\Sigma}^\circ$ is a positive definite kernel matrix. We note that the above definition is different from the popular formulation [3] or [4] in the machine learning community. The advantage of this new notation is demonstrated by the following theorem [2].

**Theorem 1.** *Let $\mathbf{A} \sim \mathcal{IW}_m(\delta, \mathbf{K})$, $\mathbf{A} \in \mathbb{R}_+$, $\mathbf{K} \in \mathbb{R}_+$, and $\mathbf{A}$ and $\mathbf{K}$ be partitioned as*

$$\mathbf{A} = \begin{bmatrix} \mathbf{A}_{11}, \mathbf{A}_{12} \\ \mathbf{A}_{21}, \mathbf{A}_{22} \end{bmatrix}, \quad \mathbf{K} = \begin{bmatrix} \mathbf{K}_{11}, \mathbf{K}_{12} \\ \mathbf{K}_{21}, \mathbf{K}_{22} \end{bmatrix}$$

*where $\mathbf{A}_{11}$ and $\mathbf{K}_{11}$ are two $n \times n$ sub matrices, $n < m$, then $\mathbf{A}_{11} \sim \mathcal{IW}_n(\delta, \mathbf{K}_{11})$.*

The new formulation of inverted Wishart is *consistent under marginalization*. Therefore, similar to the way of deriving GPs from Gaussian distributions, we define a distribution of infinite-dimensional kernel functions, denoted by $\Sigma \sim \mathcal{IW}_\infty(\delta, \Sigma^\circ)$, such that any sub kernel matrix of size $m \times m$ follows $\mathbf{\Sigma} \sim \mathcal{IW}_m(\delta, \mathbf{\Sigma}^\circ)$, where both $\Sigma$ and $\Sigma^\circ$ are positive definite kernel functions. In case when $\mathcal{U}$ and $\mathcal{V}$ are sets of entity indices, SRMs let $\Sigma^\circ$ and $\Omega^\circ$ both be Dirac delta functions, i.e., any of their sub kernel matrices is an identity matrix.

Similar to GP regression/classification, the major application of SRMs is supervised prediction based on observed relational values and input features of entities. Formally, let $\mathbf{Y}_\mathbb{I} = \{y(u, v)|(u, v) \in \mathbb{I}\}$ be the set of noisy observations, where $\mathbb{I} \subset \mathcal{U} \times \mathcal{V}$, the model aims to predict the noise-free values $Z_\mathbb{O} = \{z(u, v)|(u, v) \in \mathbb{O}\}$ on $\mathbb{O} \subset \mathcal{U} \times \mathcal{V}$. As our computation is always on a finite set containing both $\mathbb{I}$ and $\mathbb{O}$, from now on, we only consider the finite subset $\mathcal{U}_0 \times \mathcal{V}_0$, a finite support subset of $\mathcal{U} \times \mathcal{V}$ that contains $\mathbb{I} \cup \mathbb{O}$. Accordingly we let $\mathbf{\Sigma}$ be the covariance matrix of $\Sigma$ on $\mathcal{U}_0$, and $\mathbf{\Omega}$ be the covariance matrix of $\Omega$ on $\mathcal{V}_0$.

Previously a variational Bayesian method was applied to SRMs [15], which computes the maximum *a posterior* estimates of $\mathbf{\Sigma}$ and $\mathbf{\Omega}$, given $\mathbf{Y}_\mathbb{I}$, and then predicts $\mathbf{Z}_\mathbb{O}$ based on the estimated $\mathbf{\Sigma}$ and $\mathbf{\Omega}$. There are two limitations of this empirical Bayesian approach: (1) The variational method is not a fully Bayesian treatment. Ideally we wish to integrate $\mathbf{\Sigma}$ and $\mathbf{\Omega}$; (2) The more critical issue is, the algorithm has the complexity $O(m^3 + n^3)$, with $m = |\mathcal{U}_0|$ and $n = |\mathcal{V}_0|$, is not scalable to a large relational domain where $m$ or $n$ exceeds several thousands. In this paper we will introduce a fully Bayesian inference algorithm using Markov chain Monte Carlo sampling. By deriving equivalent sampling processes, we show the algorithms can be applied to a dataset, which is $10^3$ times larger than the previous work [15], and produce an excellent accuracy.

In the rest of this paper, we present our algorithms for Bayesian inference of SRMs in Section 2. Some related work is discussed in Section 3, followed by experiment results of SRMs in Section 4. Section 5 concludes.

## 2 Bayesian Models and MCMC Inference

In this paper, we tackle the scalability issue with a fully Bayesian paradigm. We estimate the expectation of $\mathbf{Z}_{\mathbb{O}}$ directly from $\mathbf{Y}_{\mathbb{I}}$ using Markov-chain Monte Carlo (MCMC) algorithm (specifically, Gibbs sampling), instead of evaluating that from estimated $\mathbf{\Sigma}$ or $\mathbf{\Omega}$. Our contribution is in how to make the MCMC inference more efficient for large scale data.

We first introduce some necessary notation here. Bold capital letters, e.g. $\mathbf{X}$, indicate matrices. $\mathbf{I}_{(m)}$ is an identity matrix of size $m \times m$. $\mathcal{N}_d$, $\mathcal{N}_{m,d}$, $\mathcal{IW}_m$, $\chi^{-2}$ are the multivariate normal distribution, the matrix-variate normal distribution, the inverse-Wishart distribution, and the inverse chi-square distribution, respectively.

### 2.1 Models with Non-informative Priors

Let $r = |\mathbb{I}|$, $m = |\mathcal{U}_0|$ and $n = |\mathcal{V}_0|$. It is assumed that $d \ll \min(m, n)$, and the observed set, $\mathbb{I}$, is sparse, i.e. $r \ll mn$. First, we consider the case of $\mathbf{\Sigma}^\circ = \alpha\mathbf{I}_{(m)}$ and $\mathbf{\Omega}^\circ = \beta\mathbf{I}_{(n)}$. Let $\{f_k\}$ on $\mathcal{U}_0$ denoted by matrix variate $\mathbf{F}$ of size $m \times d$, $\{g_k\}$ on $\mathcal{V}_0$ denoted by matrix variate $\mathbf{G}$ of size $n \times d$. Then the generative model is written as Model 2 and depicted in Figure 1.

**Model 2.** *The generative model of a matrix-variate SRM:*

1. *Draw $\mathbf{\Sigma} \sim \mathcal{IW}_m(\delta, \alpha\mathbf{I}_{(m)})$ and $\mathbf{\Omega} \sim \mathcal{IW}_n(\delta, \beta\mathbf{I}_{(n)})$;*

2. *Draw $\mathbf{F}|\mathbf{\Sigma} \sim \mathcal{N}_{m,d}(0, \mathbf{\Sigma} \otimes \mathbf{I}_{(d)})$ and $\mathbf{G}|\mathbf{\Omega} \sim \mathcal{N}_{n,d}(0, \mathbf{\Omega} \otimes \mathbf{I}_{(d)})$;*

3. *Draw $s^2 \sim \chi^{-2}(\nu, \sigma^2)$ ;*

4. *Draw $\mathbf{Y}|\mathbf{F}, \mathbf{G}, s^2 \sim \mathcal{N}_{m,n}(\mathbf{Z}, s^2\mathbf{I}_{(m)} \otimes \mathbf{I}_{(n)})$, where $\mathbf{Z} = \mathbf{F}\mathbf{G}^\top$.*

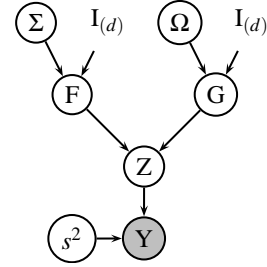

where $\mathcal{N}_{m,d}$ is the matrix-variate normal distribution of size $m \times d$; $\alpha$, $\beta$, $\delta$, $\nu$ and $\sigma^2$ are scalar parameters of the model. A slight difference

Figure 1: Model 2

between this finite model and Model 1 is that the coefficient $1/\sqrt{d}$ is ignored for simplicity because this coefficient can be absorbed by $\alpha$ or $\beta$.

As we can explicitly compute $\Pr(\mathbf{\Sigma}|\mathbf{F})$, $\Pr(\mathbf{\Omega}|\mathbf{G})$, $\Pr(\mathbf{F}|\mathbf{Y}_{\mathbb{I}}, \mathbf{G}, \mathbf{\Sigma}, s^2)$, $\Pr(\mathbf{G}|\mathbf{Y}_{\mathbb{I}}, \mathbf{F}, \mathbf{\Omega}, s^2)$, $\Pr(s^2|\mathbf{Y}_{\mathbb{I}}, \mathbf{F}, \mathbf{G})$, we can apply Gibbs sampling algorithm to compute $\mathbf{Z}_{\mathbb{O}}$. However, the computational time complexity is at least $O(m^3 + n^3)$, which is not practical for large scale data.

### 2.2 Gibbs Sampling Method

To overcome the inefficiency in sampling large covariance matrices, we rewrite the sampling process using the property of Theorem 2 to take the advantage of $d \ll \min(m, n)$.

**Theorem 2.** *If*

1. *$\mathbf{\Sigma} \sim \mathcal{IW}_m(\delta, \alpha\mathbf{I}_{(m)})$ and $\mathbf{F}|\mathbf{\Sigma} \sim \mathcal{N}_{m,d}(0, \mathbf{\Sigma} \otimes \mathbf{I}_{(d)})$,*

2. *$\mathbf{K} \sim \mathcal{IW}_d(\delta, \alpha\mathbf{I}_{(d)})$ and $\mathbf{H}|\mathbf{K} \sim \mathcal{N}_{m,d}(0, \mathbf{I}_{(m)} \otimes \mathbf{K})$,*

*then, matrix variates, $\mathbf{F}$ and $\mathbf{H}$, have the same distribution.*

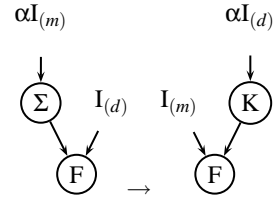

Figure 2: Theorem 2

*Proof sketch.* Matrix variate $\mathbf{F}$ follows a matrix variate $t$ distribution, $t(\delta, 0, \alpha\mathbf{I}_{(m)}, \mathbf{I}_{(d)})$, which is written as

$$p(\mathbf{F}) \propto |\mathbf{I}_{(m)} + (\alpha\mathbf{I}_{(m)})^{-1}\mathbf{F}(\mathbf{I}_{(d)})^{-1}\mathbf{F}^\top|^{-\frac{1}{2}(\delta+m+d-1)} = |\mathbf{I}_{(m)} + \alpha^{-1}\mathbf{F}\mathbf{F}^\top|^{-\frac{1}{2}(\delta+m+d-1)}$$

Matrix variate $\mathbf{H}$ follows a matrix variate $t$ distribution, $t(\delta, 0, \mathbf{I}_{(m)}, \alpha\mathbf{I}_{(d)})$, which can be written as

$$p(\mathbf{H}) \propto |\mathbf{I}_{(m)} + (\mathbf{I}_{(m)})^{-1}\mathbf{H}(\alpha\mathbf{I}_{(d)})^{-1}\mathbf{H}^\top|^{-\frac{1}{2}(\delta+m+d-1)} = |\mathbf{I}_{(m)} + \alpha^{-1}\mathbf{H}\mathbf{H}^\top|^{-\frac{1}{2}(\delta+m+d-1)}$$

Thus, matrix variates, $\mathbf{F}$ and $\mathbf{H}$, have the same distribution. □

This theorem allows us to sample a smaller covariance matrix $\mathbf{K}$ of size $d \times d$ on the column side instead of sampling a large covariance matrix $\mathbf{\Sigma}$ of size $m \times m$ on the row side. The translation is depicted in Figure 2. This theorem applies to $\mathbf{G}$ as well, thus we rewrite the model as Model 3 (or Figure 3). A similar idea was used in our previous work [16].

**Model 3.** *The alternative generative model of a matrix-variate SRM:*

1. *Draw $\mathbf{K} \sim \mathcal{IW}_d(\delta, \alpha\mathbf{I}_{(d)})$ and $\mathbf{R} \sim \mathcal{IW}_d(\delta, \beta\mathbf{I}_{(d)})$;*

2. *Draw $\mathbf{F}|\mathbf{K} \sim \mathcal{N}_{m,d}(0, \mathbf{I}_{(m)} \otimes \mathbf{K})$, and $\mathbf{G}|\mathbf{R} \sim \mathcal{N}_{n,d}(0, \mathbf{I}_{(n)} \otimes \mathbf{R})$,*

3. *Draw $s^2 \sim \chi^{-2}(\nu, \sigma^2)$ ;*

4. *Draw $\mathbf{Y}|\mathbf{F}, \mathbf{G}, s^2 \sim \mathcal{N}_{m,n}(\mathbf{Z}, s^2\mathbf{I}_{(m)} \otimes \mathbf{I}_{(n)})$, where $\mathbf{Z} = \mathbf{F}\mathbf{G}^\top$.*

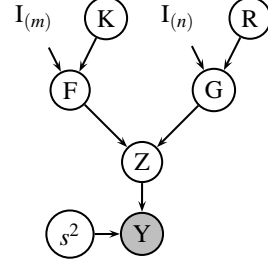

Let column vector $\mathbf{f}_i$ be the $i$-th row of matrix $\mathbf{F}$, and column vector $\mathbf{g}_j$ be the $j$-th row of matrix $\mathbf{G}$. In Model 3, $\{\mathbf{f}_i\}$ are independent given $\mathbf{K}$, $\mathbf{G}$ and $s^2$. Similar independence applies to $\{\mathbf{g}_j\}$ as well. The conditional posterior distribution of $\mathbf{K}, \mathbf{R}, \{\mathbf{f}_i\}, \{\mathbf{g}_j\}$ and $s^2$ can be easily computed, thus the Gibbs sampling for SRM is named BSRM (for Bayesian SRM).

Figure 3: Model 3

We use Gibbs sampling to compute the mean of $\mathbf{Z}_{\mathbb{O}}$, which is derived from the samples of $\mathbf{F}\mathbf{G}^\top$. Because of the sparsity of $\mathbb{I}$, each iteration in this sampling algorithm can be computed in $O(d^2 r + d^3(m+n))$ time complexity[2], which is a dramatic reduction from the previous time complexity $O(m^3 + n^3)$ .

## 2.3 Models with Informative Priors

An important characteristic of SRMs is that it allows the inclusion of certain prior knowledge of entities into the model. Specifically, the prior information is encoded as the prior covariance parameters, i.e. $\mathbf{\Sigma}^\circ$ and $\mathbf{\Omega}^\circ$. In the general case, it is difficult to run sampling process due to the size of $\mathbf{\Sigma}^\circ$ and $\mathbf{\Omega}^\circ$. We assume that $\mathbf{\Sigma}^\circ$ and $\mathbf{\Omega}^\circ$ have a special form, i.e. $\mathbf{\Sigma}^\circ = \mathbf{F}^\circ(\mathbf{F}^\circ)^\top + \alpha\mathbf{I}_{(m)}$, where $\mathbf{F}^\circ$ is an $m \times p$ matrix, and $\mathbf{\Omega}^\circ = \mathbf{G}^\circ(\mathbf{G}^\circ)^\top + \beta\mathbf{I}_{(n)}$, where $\mathbf{G}^\circ$ is an $n \times q$ matrix, and the magnitude of $p$ and $q$ is about the same as or less than that of $d$. This prior knowledge can be obtained from some additional *features* of entities.

Although such an informative $\mathbf{\Sigma}^\circ$ prevents us from directly sampling each row of $\mathbf{F}$ independently, as we do in Model 3, we can expand matrix $\mathbf{F}$ of size $m \times d$ to $(\mathbf{F}, \mathbf{F}^\circ)$ of size $m \times (d+p)$, and derive an equivalent model, where rows of $\mathbf{F}$ are conditionally independent given $\mathbf{F}^\circ$. Figure 4 illustrates this transformation.

**Theorem 3.** *Let $\delta > p$, $\mathbf{\Sigma}^\circ = \mathbf{F}^\circ(\mathbf{F}^\circ)^\top + \alpha\mathbf{I}_{(m)}$, where $\mathbf{F}^\circ$ is an $m \times p$ matrix. If*

1. $\mathbf{\Sigma} \sim \mathcal{IW}_m(\delta, \mathbf{\Sigma}^\circ)$ *and* $\mathbf{F}|\mathbf{\Sigma} \sim \mathcal{N}_{m,d}(\mathbf{0}, \mathbf{\Sigma} \otimes \mathbf{I}_{(d)})$,

2. $\mathbf{K} = \begin{pmatrix} \mathbf{K}_{11} & \mathbf{K}_{12} \\ \mathbf{K}_{21} & \mathbf{K}_{22} \end{pmatrix} \sim \mathcal{IW}_{d+p}(\delta - p, \alpha\mathbf{I}_{(d+p)})$ *and* $\mathbf{H}|\mathbf{K} \sim \mathcal{N}_{m,d}(\mathbf{F}^\circ\mathbf{K}_{22}^{-1}\mathbf{K}_{21}, \mathbf{I}_{(m)} \otimes \mathbf{K}_{11\cdot2})$,

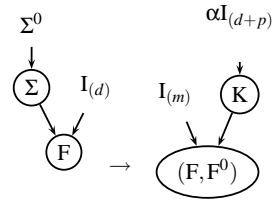

*where $\mathbf{K}_{11\cdot2} = \mathbf{K}_{11} - \mathbf{K}_{12}\mathbf{K}_{22}^{-1}\mathbf{K}_{21}$, then $\mathbf{F}$ and $\mathbf{H}$ have the same distribution.*

Figure 4: Theorem 3

*Proof sketch.* Consider the distribution

$$(\mathbf{H}_1, \mathbf{H}_2)|\mathbf{K} \sim \mathcal{N}_{m,d+p}(\mathbf{0}, \mathbf{I}_{(m)} \otimes \mathbf{K}). \tag{1}$$

Because $\mathbf{H}_1|\mathbf{H}_2 \sim \mathcal{N}_{m,d}(\mathbf{H}_2\mathbf{K}_{22}^{-1}\mathbf{K}_{21}, \mathbf{I}_{(m)} \otimes \mathbf{K}_{11\cdot2})$, $p(\mathbf{H}) = p(\mathbf{H}_1|\mathbf{H}_2 = \mathbf{F}^\circ)$. On the other hand, we have a matrix-variate $t$ distribution, $(\mathbf{H}_1, \mathbf{H}_2) \sim t_{m,d+p}(\delta - p, \mathbf{0}, \alpha\mathbf{I}_{(m)}, \mathbf{I}_{(d+p)})$. By Theorem 4.3.9 in [4], we have $\mathbf{H}_1|\mathbf{H}_2 \sim t_{m,d}(\delta, \mathbf{0}, \alpha\mathbf{I}_{(m)} + \mathbf{H}_2\mathbf{H}_2^\top, \mathbf{I}_{(d)}) = t_{m,d}(\delta, \mathbf{0}, \mathbf{\Sigma}^\circ, \mathbf{I}_{(d)})$, which implies $p(\mathbf{F}) = p(\mathbf{H}_1|\mathbf{H}_2 = \mathbf{F}^\circ) = p(\mathbf{H})$. $\square$

The following corollary allows us to compute the posterior distribution of $\mathbf{K}$ efficiently.

**Corollary 4.** $\mathbf{K}|\mathbf{H} \sim \mathcal{IW}_{d+p}(\delta + m, \alpha\mathbf{I}_{(d+p)} + (\mathbf{H}, \mathbf{F}^\circ)^\top (\mathbf{H}, \mathbf{F}^\circ))$.

*Proof sketch.* Because normal distribution and inverse Wishart distribution are conjugate, we can derive the posterior distribution $\mathbf{K}$ from Eq. (1). $\qquad\square$

Thus, we can explicitly sample from the conditional posterior distributions, as listed in Algorithm 1 (BSRM/F for BSRM with features) in Appendix. We note that when $p = q = 0$, Algorithm 1 (BSRM/F) reduces to the exact algorithm for BSRM. Each iteration in this sampling algorithm can be computed in $O(d^2r + d^3(m + n) + dpm + dqn)$ time complexity.

### 2.4 Unblocking for Sampling Implementation

Blocking Gibbs sampling technique is commonly used to improve the sampling efficiency by reducing the sample variance according to the Rao-Blackwell theorem (c.f. [9]). However, blocking Gibbs sampling is not necessary to be computationally efficient. To improve the computational efficiency of Algorithm 1, we use unblocking sampling to reduce the major computational cost is Step 2 and Step 4. We consider sampling each element of $\mathbf{F}$ conditionally. The sampling process is written as Step 4 and Step 9 of Algorithm 2, which is called BSRM/F with conditional Gibss sampling. We can reduce the computational cost of each iteration to $O(dr + d^2(m + n) + dpm + dqn)$, which is comparable to other low-rank matrix factorization approaches. Though such a conditional sampling process increases the sample variance comparing to Algorithm 1, we can afford more samples within a given amount of time due to its faster speed. Our experiments show that the overall computational cost of Algorithm 2 is usually less than that of Algorithm 1 when achieving the same accuracy. Additionally, since $\{\mathbf{f}_i\}$ are independent, we can parallelize the *for* loops in Step 4 and Step 9 of Algorithm 2.

## 3 Related Work

SRMs fall into a class of statistical latent-variable relational models that explain relations by latent factors of entities. Recently a number of such models were proposed that can be roughly put into two groups, depending on whether the latent factors are continuous or discrete: (1) Discrete latent-state relational models: a large body of research infers latent classes of entities and explains the entity relationship by the probability conditioned on the joint state of participating entities, e.g., [6, 14, 7, 1]. In another work [10], binary latent factors are modeled; (2) Continuous latent-variable relational models: many such models assume relational data underlain by multiplicative effects between latent variables of entities, e.g. [5]. A simple example is matrix factorization, which recently has become very popular in collaborative filtering applications, e.g., [12, 8, 13].

The latest Bayesian probabilistic matrix factorization [13] reported the state-of-the-art accuracy of matrix factorization on Netflix data. Interestingly, the model turns out to be similar to our Model 3 under the non-informative prior. This paper reveals the equivalence between different models and offers a more general Bayesian framework that allows informative priors from entity features to play a role. The framework also generalizes Gaussian processes [11] to a relational domain, where a nonparametric prior for stochastic relational processes is described.

## 4 Experiments

**Synthetic data:** We compare BSRM under noninformative priors against two other algorithms: the fast max-margin matrix factorization (fMMMF) in [12] with a square loss, and SRM using variational Bayesian approach (SRM-VB) in [15]. We generate a $30 \times 20$ random matrix (Figure 5(a)), then add Gaussian noise with $\sigma^2 = 0.1$ (Figure 5(b)). The root mean squared noise is 0.32. We select 70% elements as the observed data and use the rest of the elements for testing. The reconstruction matrix and root mean squared errors (RMSEs) of predictions on the test elements are shown in Figure 5(c)-5(e). BSRM outperforms the variational approach of SRMs and fMMMF. Note that because of the log-determinant penalty of the inverse Wishart prior, SRM-VB enforces the rank to be smaller, thus the result of SRM-VB looks smoother than that of BSRM.

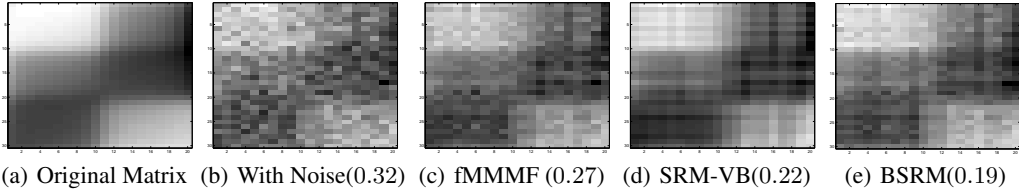

(a) Original Matrix  (b) With Noise(0.32)  (c) fMMMF (0.27)  (d) SRM-VB(0.22)  (e) BSRM(0.19)

Figure 5: Experiments on synthetic data. RMSEs are shown in parentheses.

|  | User Mean | Movie Mean | fMMMF [12] | VB [8] |
|---|---|---|---|---|
| RMSE | 1.425 | 1.387 | 1.186 | 1.165 |
| MAE | 1.141 | 1.103 | 0.943 | 0.915 |

Table 1: RMSE (root mean squared error) and MAE (mean absolute error) of the experiments on EachMovie data. All standard errors are 0.001 or less.

**EachMovie data:** We test the accuracy and the efficiency of our algorithms on EachMovie. The dataset contains $74,424$ users' $2,811,718$ ratings on $1,648$ movies, i.e. about $2.29\%$ are rated by zero-to-five stars. We put all the ratings into a matrix, and randomly select $80\%$ as observed data to predict the remaining ratings. The random selection was carried out 10 times independently. We compare our approach against several competing methods: 1) User Mean, predicting ratings by the sample mean of the same user's ratings; 2) Move Mean, predicting rating by the sample mean of ratings on the same movie; 3) fMMMF [12]; 4) VB introduced in [8], which is a probabilistic low-rank matrix factorization using variational approximation. Because of the data size, we cannot run the SRM-VB of [15]. We test the algorithms BSRM and BSRM/F, both following Algorithm 2, which run without and with features, respectively. The features used in BSRM/F are generated from the PCA result of the binary indicator matrix that indicates whether the user rates the movie. The 10 top factors of both the user side and the movie side are used as features, i.e. $p = 10$, $q = 10$. We run the experiments with different $d = 16, 32, 100, 200, 300$. The hyper parameters are set to some trivial values, $\delta = p + 1 = 11$, $\alpha = \beta = 1$, $\sigma^2 = 1$, and $\nu = 1$. The results are shown in Table 1 and 2. We find that the accuracy improves as the number of $d$ is increased. Once $d$ is greater than 100, the further improvement is not very significant within a reasonable amount of running time.

| rank ($d$) | | 16 | 32 | 100 | 200 | 300 |
|---|---|---|---|---|---|---|
| BSRM | RMSE | 1.0983 | 1.0924 | 1.0905 | 1.0903 | 1.0902 |
|  | MAE | 0.8411 | 0.8321 | 0.8335 | 0.8340 | 0.8393 |
| BSRM/F | RMSE | 1.0952 | 1.0872 | 1.0848 | 1.0846 | 1.0852 |
|  | MAE | 0.8311 | 0.8280 | 0.8289 | 0.8293 | 0.8292 |

Table 2: RMSE (root mean squared error) and MAE (mean absolute error) of experiments on EachMovie data. All standard errors are 0.001 or less.

To compare the overall computational efficiency of the two Gibbs sampling procedures, Algorithm 1 and Algorithm 2, we run both algorithms and record the running time and accuracy in RMSE. The dimensionality $d$ is set to be 100. We compute the average $\mathbf{Z}_\mathbb{O}$ and evaluate it after a certain number of iterations. The evaluation results are shown in Figure 6. We run both algorithms for 100 iterations as the burn-in period, so that we can have an independent start sample. After the burn-in period, we restart to compute the averaged $\mathbf{Z}_\mathbb{O}$ and evaluate them, therefore there are abrupt points at 100 iterations in both cases. The results show that the overall accuracy of Algorithm 2 is better at any given time.

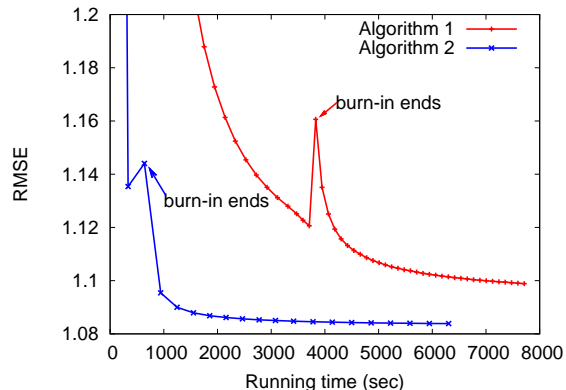

Figure 6: Time-Accuracy of Algorithm 1 and 2

**Netflix data:** We also test the algorithms on the large collection of user ratings from netflix.com. The dataset consists of $100, 480, 507$ ratings from $480, 189$ users on $17, 770$ movies. In addition, Netflix also provides a set of validation data with $1, 408, 395$ ratings. In order to evaluate the prediction accuracy, there is a test set containing $2, 817, 131$ ratings whose values are withheld and unknown for all the participants.

The features used in BSRM/F are generated from the PCA result of a binary matrix that indicates whether or not the user rated the movie. The top 30 user-side factors are used as features, none of movie-side factors are used, i.e. $p = 30$, $q = 0$. The hyper parameters are set to some trivial values, $\delta = p + 1 = 31$, $\alpha = \beta = 1$, $\sigma^2 = 1$, and $\nu = 1$. The results on the validation data are shown in Table 3. The submitted result of BSRM/F(400) achieves RMSE $0.8881$ on the test set. The running time is around 21 minutes per iteration for $400$ latent dimensions on an Intel Xeon 2GHz PC.

|  | VB[8] | BPMF [13] | BSRM 100 | BSRM 200 | BSRM 400 | BSRM/F 100 | BSRM/F 200 | BSRM/F 400 |
|---|---|---|---|---|---|---|---|---|
| RMSE | 0.9141 | 0.8920 | 0.8930 | 0.8910 | 0.8895 | 0.8926 | 0.8880 | 0.8874 |

Table 3: RMSE (root mean squared error) of experiments on Netflix data.

## 5    Conclusions

In this paper, we study the fully Bayesian inference for stochastic relational models (SRMs), for learning the real-valued relation between entities of two sets. We overcome the scalability issue by transforming SRMs into equivalent models, which can be efficiently sampled. The experiments show that the fully Bayesian inference outperforms the previously used variational Bayesian inference on SRMs. In addition, some techniques for efficient computation in this paper can be applied to other large-scale Bayesian inferences, especially for models involving inverse-Wishart distributions.

**Acknowledgment:** We thank the reviewers and Sarah Tyler for constructive comments.

## Footnotes

[1]We denote an $n$ dimensional Gaussian distribution with a covariance matrix $\mathbf{\Sigma}$ by $\mathcal{N}_n(\mathbf{0}, \mathbf{\Sigma})$. Then $\mathcal{N}_{\infty}(0, \Sigma)$ explicitly indicates that a GP follows an "infinite dimensional" Gaussian distribution.

[2] $|\mathbf{Y} - \mathbf{F}\mathbf{G}^\top|_{\mathbb{I}}^2$ can be efficiently computed in $O(dr)$ time.

## References

[1] E. Airodi, D. Blei, S. Fienberg, and E. P. Xing. Mixed membership stochastic blockmodels. In *Journal of Machine Learning Research*, 2008.

[2] A. P. Dawid. Some matrix-variate distribution theory: notational considerations and a Bayesian application. *Biometrika*, 68:265–274, 1981.

[3] A. Gelman, J. B. Carlin, H. S. Stern, and D. B. Rubin. *Bayesian Data Analysis*. Chapman & Hall, New York, 1995.

[4] A. K. Gupta and D. K. Nagar. *Matrix Variate Distributions*. Chapman & Hall/CRC, 2000.

[5] P. Hoff. Multiplicative latent factor models for description and prediction of social networks. *Computational and Mathematical Organization Theory*, 2007.

[6] T. Hofmann. Latent semantic models for collaborative filtering. *ACM Trans. Inf. Syst.*, 22(1):89–115, 2004.

[7] C. Kemp, J. B. Tenenbaum, T. L. Griffiths, T. Yamada, and N. Ueda. Learning systems of concepts with an infinite relational model. In *Proceedings of the 21st National Conference on Artificial Intelligence (AAAI)*, 2006.

[8] Y. J. Lim and Y. W. Teh. Variational Bayesian approach to movie rating prediction. In *Proceedings of KDD Cup and Workshop*, 2007.

[9] J. S. Liu. *Monte Carlo Strategies in Scientific Computing*. Springer, 2001.

[10] E. Meeds, Z. Ghahramani, R. Neal, and S. T. Roweis. Modeling dyadic data with binary latent factors. In *Advances in Neural Information Processing Systems 19*, 2007.

[11] C. E. Rasmussen and C. K. I. Williams. *Gaussian Processes for Machine Learning*. MIT Press, 2006.

[12] J. D. M. Rennie and N. Srebro. Fast maximum margin matrix factorization for collaborative prediction. In *ICML*, 2005.

[13] R. Salakhutdinov and A. Mnih. Bayeisna probabilistic matrix factorization using Markov chain Monte Carlo. In *The 25th International Conference on Machine Learning*, 2008.

[14] Z. Xu, V. Tresp, K. Yu, and H.-P. Kriegel. Infinite hidden relational models. In *Proceedings of the 22nd International Conference on Uncertainty in Artificial Intelligence (UAI)*, 2006.

[15] K. Yu, W. Chu, S. Yu, V. Tresp, and Z. Xu. Stochastic relational models for discriminative link prediction. In *Advances in Neural Information Processing Systems 19 (NIPS)*, 2006.

[16] S. Zhu, K. Yu, and Y. Gong. Predictive matrix-variate t models. In J. Platt, D. Koller, Y. Singer, and S. Roweis, editors, *NIPS '07: Advances in Neural Information Processing Systems 20*, pages 1721–1728. MIT Press, Cambridge, MA, 2008.

## Appendix

Before presenting the algorithms, we introduce the necessary notation. Let $\mathbb{I}^i = \{j|(i,j) \in \mathbb{I}\}$ and $\mathbb{I}_j = \{i|(i,j) \in \mathbb{I}\}$. A matrix with subscripts indicates its submatrix, which consists its entries at the given indices in the subscripts, for example, $\mathbf{X}_{\mathbb{I}_j,j}$ is a subvector of the $j$-th column of $\mathbf{X}$ whose row indices are in set $\mathbb{I}_j$, $\mathbf{X}_{\cdot,j}$ is the $j$-th column of $\mathbf{X}$ ($\cdot$ indicates the full set). $X_{i,j}$ denotes the $(i,j)$-th entry of $\mathbf{X}$. $|\mathbf{X}|_{\mathbb{I}}^2$ is the squared sum of elements in set $\mathbb{I}$, i.e. $\sum_{(i,j) \in \mathbb{I}} X_{i,j}^2$. We fill the unobserved elements in $\mathbf{Y}$ with 0 for simplicity in notation

---

**Algorithm 1** BSRM/F: Gibbs sampling for SRM with features

---

1: Draw $\mathbf{K} \sim \mathcal{IW}_{d+p}(\delta + m, \alpha\mathbf{I}_{(d+p)} + (\mathbf{F}, \mathbf{F}^\circ)^\top(\mathbf{F}, \mathbf{F}^\circ))$;

2: For each $i \in \mathcal{U}_0$, draw $\mathbf{f}_i \sim \mathcal{N}_d(\mathbf{K}_{(i)}(s^{-2}\mathbf{G}^\top\mathbf{Y}_{i,\cdot}^\top + \mathbf{K}_{11\cdot2}^{-1}\mathbf{K}_{12}\mathbf{K}_{22}^{-1}\mathbf{f}_i^\circ), \mathbf{K}_{(i)})$,

　　where $\mathbf{K}_{(i)} = \left(s^{-2}(\mathbf{G}_{\mathbb{I}^i,\cdot})^\top\mathbf{G}_{\mathbb{I}^i,\cdot} + \mathbf{K}_{11\cdot2}^{-1}\right)^{-1}$;

3: Draw $\mathbf{R} \sim \mathcal{IW}_{d+q}(\delta + n, \beta\mathbf{I}_{(d+q)} + (\mathbf{G}, \mathbf{G}^\circ)^\top(\mathbf{G}, \mathbf{G}^\circ))$;

4: For each $j \in \mathcal{V}_0$, draw $\mathbf{g}_j \sim \mathcal{N}_d(\mathbf{R}_{(j)}(s^{-2}\mathbf{F}^\top\mathbf{Y}_{\cdot,j} + \mathbf{R}_{11\cdot2}^{-1}\mathbf{R}_{12}\mathbf{R}_{22}^{-1}\mathbf{g}_j^\circ), \mathbf{R}_{(j)})$,

　　where $\mathbf{R}_{(j)} = \left(s^{-2}(\mathbf{F}_{\mathbb{I}_j,\cdot})^\top\mathbf{F}_{\mathbb{I}_j,\cdot} + \mathbf{R}_{11\cdot2}^{-1}\right)^{-1}$;

5: Draw $s^2 \sim \chi^{-2}(\nu + r, \sigma^2 + |\mathbf{Y} - \mathbf{F}\mathbf{G}^\top|_{\mathbb{I}}^2)$.

---

**Algorithm 2** BSRM/F: Conditional Gibbs sampling for SRM with features

---

1: $\Delta_{i,j} \leftarrow Y_{i,j} - \sum_k F_{i,k}G_{j,k}$, for $(i,j) \in \mathbb{I}$;

2: Draw $\mathbf{\Phi} \sim \mathcal{W}_{d+p}(\delta + m + d + p - 1, (\alpha\mathbf{I}_{(d+p)} + (\mathbf{F}, \mathbf{F}^\circ)^\top(\mathbf{F}, \mathbf{F}^\circ))^{-1})$;

3: **for** each $(i,k) \in \mathcal{U}_0 \times \{1, \cdots, d\}$ **do**

4: 　　Draw $f \sim \mathcal{N}_1(\phi^{-1}(s^{-2}\mathbf{\Delta}_{i,\mathbb{I}^i}\mathbf{G}_{\mathbb{I}^i,k} - \mathbf{F}_{i,\cdot}\mathbf{\Phi}_{\cdot,k}), \phi^{-1})$, where $\phi = s^{-2}(\mathbf{G}_{\mathbb{I}^i,k})^\top\mathbf{G}_{\mathbb{I}^i,k} + \Phi_{k,k}$;

5: 　　Update $F_{i,k} \leftarrow F_{i,k} + f$, and $\Delta_{i,j} \leftarrow \Delta_{i,j} - fG_{j,k}$, for $j \in \mathbb{I}^i$;

6: **end for**

7: Draw $\mathbf{\Psi} \sim \mathcal{W}_{d+q}(\delta + n + d + q - 1, (\beta\mathbf{I}_{(d+q)} + (\mathbf{G}, \mathbf{G}^\circ)^\top(\mathbf{G}, \mathbf{G}^\circ))^{-1})$;

8: **for** each $(j,k) \in \mathcal{V}_0 \times \{1, \cdots, d\}$ **do**

9: 　　Draw $g \sim \mathcal{N}_1(\psi^{-1}(s^{-2}\mathbf{\Delta}_{\mathbb{I}_j,j}^\top\mathbf{F}_{\mathbb{I}_j,k} - \mathbf{G}_{j,\cdot}\mathbf{\Psi}_{\cdot,k}), \psi^{-1})$, where $\psi = s^{-2}(\mathbf{F}_{\mathbb{I}_j,k})^\top\mathbf{F}_{\mathbb{I}_j,k} + \Psi_{k,k}$;

10: 　Update $G_{j,k} \leftarrow G_{j,k} + g$ and $\Delta_{i,j} \leftarrow \Delta_{i,j} - gF_{i,k}$, for $i \in \mathbb{I}_j$;

11: **end for**

12: Draw $s^2 \sim \chi^{-2}(\nu + r, \sigma^2 + |\mathbf{\Delta}|_{\mathbb{I}}^2)$.

---

